# An Oracle Inequality for Clipped Regularized Risk Minimizers

**Ingo Steinwart, Don Hush, and Clint Scovel**
Modelling, Algorithms and Informatics Group, CCS-3
Los Alamos National Laboratory
Los Alamos, NM 87545
{ingo,dhush,jcs}@lanl.gov

## Abstract

We establish a general oracle inequality for clipped approximate minimizers of regularized empirical risks and apply this inequality to support vector machine (SVM) type algorithms. We then show that for SVMs using Gaussian RBF kernels for classification this oracle inequality leads to learning rates that are faster than the ones established in [9]. Finally, we use our oracle inequality to show that a simple parameter selection approach based on a validation set can yield the same fast learning rates without knowing the noise exponents which were required to be known a-priori in [9].

## 1 Introduction

The theoretical understanding of support vector machines (SVMs) and related kernel-based methods has been substantially improved in recent years. For example using Talagrand's concentration inequality and local Rademacher averages it has recently been shown that SVMs for classification can learn with rates up to $n^{-1}$ under somewhat realistic assumptions on the data-generating distribution (see [9, 11] and the related work [2]). However, the so-called "shrinking technique" of [9, 11] for establishing such rates, requires the free parameters to be chosen *a-priori*, and in addition, the optimal values of these parameters depend on features of the data-generating distribution which are typically unknown. Consequently, [9, 11] do not provide a practical method for learning with fast rates. On the other hand, the oracle inequality in [2] only holds for distributions having Tsybakov noise exponent $\infty$, and hence it describes a situation which is rarely met in practice.

The goal of this work is to overcome these shortcomings by establishing a general oracle inequality (see Theorem 3.1) for regularized empirical risk minimizers. The key ingredient of this oracle inequality is the observation that for most commonly used loss functions it is possible to "clip" the decision function of the algorithm before beginning with the theoretical analysis. In addition, a careful choice of the weighted empirical process Talagrand's inequality is applied to, makes the "shrinking technique" superfluous. Finally, by explicitly dealing with $\epsilon$-approximate minimizers of the regularized risk our results also apply to actual SVM algorithms.

With the help of the general oracle inequality we then establish an oracle inequality for SVM type algorithms (see Theorem 2.1) as well as a simple oracle inequality for model selection (see Theorem 4.2). For the former, we show that it leads to improved rates for e.g. binary classification under the assumptions considered in [9] and a-priori known noise exponents. Using the model selection theorem we then show how our new oracle inequality for SVMs can be used to analyze a simple parameter selection procedure based on a validation set that achieves the same learning rates without prior knowledge on the noise exponents.

The rest of this work is organized as follows: In Section 2 we present our oracle inequality for SVM type algorithms. We then discuss its implications and analyze the simple parameter selection

procedure when using Gaussian RBF kernels. In Section 3 we then present and prove the general oracle inequality. The proof of Theorem 2.1 as well as the oracle inequality for model selection can be found in Section 4.

## 2   Main Results

Throughout this work we assume that $X$ is compact metric space, $Y \subset [-1, 1]$ is compact, $P$ is a Borel probability measure on $X \times Y$, and $\mathcal{F}$ is a set of functions over $X$ such that $0 \in \mathcal{F}$. Often $\mathcal{F}$ is a reproducing kernel Hilbert space (RKHS) $H$ of continuous functions over $X$ with closed unit ball $B_H$. It is well-known that $H$ can then be continuously embedded into the space of continuous functions $C(X)$ equipped with the usual maximum-norm $\|.\|_\infty$. In order to avoid constants we always assume that this embedding has norm 1, i.e. $\|.\|_\infty \leq \|.\|_H$. Furthermore, $L : Y \times \mathbb{R} \to [0, \infty)$ always denotes a continuous function which is convex in its second variable such that $L(y, 0) \leq 1$. The functions $L$ will serve as loss functions and consequently let us recall that the associated $L$-risk of a measurable function $f : X \to \mathbb{R}$ is defined by

$$\mathcal{R}_{L,P}(f) = \mathbb{E}_{(x,y)\sim P} L\big(y, f(x)\big).$$

Note that the assumption $L(y, 0) \leq 1$ immediately gives $\mathcal{R}_{L,P}(0) \leq 1$. Furthermore, the minimal $L$-risk is denoted by $\mathcal{R}^*_{L,P}$, i.e. $\mathcal{R}^*_{L,P} = \inf\{\mathcal{R}_{L,P}(f) \,|\, f : X \to \mathbb{R} \text{ measurable}\}$, and a function attaining this infimum is denoted by $f^*_{L,P}$. We always assume that such an $f^*_{L,P}$ exists.

The learning schemes we are mainly interested in are based on an optimization problem of the form

$$f_{P,\lambda} := \arg\min_{f \in H} \left( \lambda \|f\|_H^2 + \mathcal{R}_{L,P}(f) \right), \tag{1}$$

where $\lambda > 0$. Note that if we identify a training set $T = ((x_1, y_1), \ldots, (x_n, y_n)) \in (X \times Y)^n$ with its empirical measure, then $f_{T,\lambda}$ denotes the empirical estimators of the above learning scheme. Obviously, support vector machines (see e.g. [5]) and regularization networks (see e.g. [7]) are both learning algorithms which fall into the above category. One way to describe the approximation error of these learning schemes is the *approximation error function*

$$a(\lambda) := \lambda \|f_{P,\lambda}\|_H^2 + \mathcal{R}_{L,P}(f_{P,\lambda}) - \mathcal{R}^*_{L,P}, \qquad \lambda > 0,$$

which has been discussed in some detail in [10]. Furthermore in order to deal with the complexity of the used RKHSs let us recall that for a subset $A \subset E$ of a Banach space $E$ the *covering numbers* are defined by

$$\mathcal{N}(A, \varepsilon, E) := \min\left\{ n \geq 1 : \exists x_1, \ldots, x_n \in E \text{ with } A \subset \bigcup_{i=1}^{n} (x_i + \varepsilon B_E) \right\}, \qquad \varepsilon > 0,$$

where $B_E$ denotes the closed unit ball of $E$. Given a finite sequence $T = ((x_1, y_1), \ldots, (x_n, y_n)) \in (X \times Y)^n$ we write $T_X := (x_1, \ldots, x_n)$. For our main results we are particularly interested in covering numbers in the Hilbert space $L_2(T_X)$ which consists of all equivalence classes of functions $f : X \times Y \to \mathbb{R}$ and which is equipped with the norm

$$\|f\|_{L_2(T_X)} := \left( \frac{1}{n} \sum_{i=1}^{n} |f(x_i)|^2 \right)^{\frac{1}{2}}. \tag{2}$$

In other words, $L_2(T_X)$ is a $L_2$-space with respect to the empirical measure of $(x_1, \ldots, x_n)$.

Learning schemes of the form (1) typically produce functions $f_{P,\lambda}$ with $\lim_{\lambda \to 0} \|f_{P,\lambda}\|_\infty = \infty$ (see e.g. [10] for a precise statement). Unfortunately, this behaviour has a serious negative impact on the learning rates when directly employing standard tool's such as Hoeffding's, Bernstein's or Talagrand's inequality. On the other hand, when dealing with e.g. the hinge loss it is obvious that clipping the function $f_{P,\lambda}$ at $-1$ and $1$ does not worsen the corresponding risks. Following this simple observation we will consider loss functions $L$ that satisfy the *clipping condition*

$$L(y, t) \geq \begin{cases} L(y, 1) & \text{if } t \geq 1 \\ L(y, -1) & \text{if } t \leq -1, \end{cases} \tag{3}$$

for all $y \in Y$. Recall that this type of loss function was already considered in [4, 11], but the clipping idea actually goes back to [1]. Moreover, it is elementary to check that most commonly used loss functions including the hinge loss and the least squares loss satisfy (3). Given a function $f : X \to \mathbb{R}$ we now define its *clipped version* $\hat{f} : X \to [-1, 1]$ by

$$\hat{f}(x) := \begin{cases} 1 & \text{if } f(x) > 1 \\ f(x) & \text{if } f(x) \in [-1, 1] \\ -1 & \text{if } f(x) < -1 \,. \end{cases}$$

It is clear from (3) that we always have $L(y, \hat{f}(x)) \leq L(y, f(x))$ and consequently we obtain $\mathcal{R}_{L,P}(\hat{f}) \leq \mathcal{R}_{L,P}(f)$ for all distributions $P$. Finally, we also need the following Lipschitz condition

$$|L|_1 := \sup_{y \in Y, -1 \leq t_1, t_2 \leq 1} \frac{|L(y, t_1) - L(y, t_2)|}{|t_1 - t_2|} \leq 2. \tag{4}$$

With the help of these definitions we can now state our main result which establishes an oracle inequality for clipped versions of $f_{T,\lambda}$:

**Theorem 2.1** *Let $P$ be a distribution on $X \times Y$ and let $L$ be a loss function which satisfies (3) and (4). Let $H$ be a RKHS of continuous functions on $X$. For a fixed element $f_0 \in H$ we define*

$$\begin{aligned} a(f_0) &:= \lambda \|f_0\|_H^2 + \mathcal{R}_{L,P}(f_0) - \mathcal{R}_{L,P}^* \\ B(f_0) &:= \sup_{x \in X, y \in Y} \left| L(y, f_0(x)) \right| . \end{aligned} \tag{5}$$

*In addition, we assume that we have a variance bound of the form*

$$\mathbb{E}_P \left( L \circ \hat{f} - L \circ f_{L,P}^* \right)^2 \leq v \left( \mathbb{E}_P (L \circ \hat{f} - L \circ f_{L,P}^*) \right)^\vartheta \tag{6}$$

*for constants $v \geq 1$, $\vartheta \in [0, 1]$ and all measurable $f : X \to \mathbb{R}$. Moreover, suppose that $H$ satisfies*

$$\sup_{T \in (X \times Y)^n} \log \mathcal{N} \left( B_H, \varepsilon, L_2(T_X) \right) \leq a \varepsilon^{-2p}, \qquad \varepsilon > 0, \tag{7}$$

*for some constants $p \in (0, 1)$ and $a \geq 1$. For fixed $\lambda > 0$ let $f_{T,\lambda} \in H$ be a function that minimizes $f \mapsto \lambda \|f\|_H^2 + \mathcal{R}_{L,T}(f)$ up to some $\epsilon > 0$. Then there exists a constant $K_{p,v}$ depending only on $p$ and $v$ such that for all $\tau \geq 1$ we have with probability not less than $1 - 3e^{-\tau}$ that*

$$\mathcal{R}_{L,P}(\hat{f}_{T,\lambda}) - \mathcal{R}_{L,P}^* \leq \left( \frac{K_{p,v} a}{\lambda^p n} \right)^{\frac{1}{2-\vartheta+p(\vartheta-1)}} + \frac{K_{p,v} a}{\lambda^p n} + 5 \left( \frac{32 v \tau}{n} \right)^{\frac{1}{2-\vartheta}} + \frac{140 \tau}{n} + \frac{14 B(f_0) \tau}{3n}$$

$$+ 8 a(f_0) + 4\epsilon. \tag{8}$$

The above oracle inequality has some interesting consequences as the following examples illustrate. We begin with an example that deals with a *fixed* kernel:

**Example 2.2 (Learning rates for single kernel)** *Assume that in Theorem 2.1 we have a Lipschitz continuous loss function such as the hinge loss. In addition assume that the approximation error function satisfies $a(\lambda) \leq c\lambda^\beta$, $\lambda > 0$, for some constants $c > 0$ and $\beta \in (0, 1]$. Setting $f_0 := f_{P,\lambda}$ and optimizing (8) with respect to $\lambda$ then shows that the corresponding SVM learns with rate $n^{-\gamma}$, where*

$$\gamma := \min \left\{ \frac{\beta}{\beta \left( 2 - \vartheta + p(\vartheta - 1) \right) + p}, \frac{2\beta}{\beta + 1} \right\}.$$

*Recall that this learning rate has already been obtained in [11].*

The next example investigates SVMs that use a Gaussian RBF kernel whose width may vary with the sample size:

**Example 2.3 (Classification with several Gaussian kernels)** *Let $X$ be the unit ball in $\mathbb{R}^d$ and $Y := \{-1, 1\}$. Furthermore assume that we are interested in binary classification using the hinge*

*loss and the Gaussian RKHSs $H_\sigma$ that belong to the RBF kernels $k_\sigma(x_1, x_2) := e^{-\sigma^2 \|x_1 - x_2\|^2}$ with width $\sigma > 0$. If $P$ has geometric noise exponent $\alpha \in (0, \infty)$ in the sense of [9] then it was shown in [9] that there exists a function $f_0 \in H_\sigma$ with $\|f_0\|_\infty \le 1$ and*

$$a_\sigma(f_0) \le c\left(\sigma^d \lambda + \sigma^{-\alpha d}\right), \qquad \sigma > 0,\ \lambda > 0,$$

*where $c > 0$ is a constant independent of $\lambda$ and $\sigma$. Moreover, [9, Thm. 2.1] shows that $H_\sigma$ satisfies (7) for all $p \in (0, 1)$ with*

$$a := c_{p,d,\delta}\sigma^{(1-p)(1+\delta)d}$$

*where $\delta > 0$ can be arbitrarily chosen and $c_{p,d,\delta}$ is a suitable constant. Now assume that $P$ has Tsybakov noise exponent $q \in [0, \infty]$ in the sense of [9]. It was then shown in [9] that (6) is satisfied for $\vartheta := \frac{q}{q+1}$. Minimizing (8) with respect to $\sigma$ and $\lambda$ and choosing $p$ and $\delta$ sufficiently small then yields that the corresponding SVM can learn with rate $n^{-\gamma + \varepsilon}$, where*

$$\gamma := \frac{\alpha(q+1)}{\alpha(q+2) + q + 1},$$

*and $\varepsilon > 0$ can be chosen arbitrarily small. Note that these rates are superior to those obtained in [9, Theorem 2.8].*

In the above examples the optimal parameters $\lambda$ and $\sigma$ depend on the sample size $n$ but not on the training samples $T$. However, these optimal parameters require us to know certain characteristics of the distribution such as the approximation exponent $\beta$ or the noise exponents $\alpha$ and $q$. The following example shows that the oracle inequality of Theorem 2.1 can be used to find these optimal parameters in a data-dependent fashion which does not require any a-priori knowledge:

**Example 2.4** *In this example we assume that our training set $T$ consists of $2n$ samples. We write $T_0$ for the first $n$ samples and $T_1$ for the last $n$ samples. Let $f_{T_0,\sigma,\lambda}$ be the SVM solution using a Gaussian kernel with width $\sigma$. Moreover, let $\Sigma \subset [1, n^{1/d})$ and $\Lambda \subset (0, 1]$ be finite sets with cardinality $m_\Sigma$ and $m_\Lambda$, respectively. Under the assumptions of Example 2.3 the oracle inequality (8) then shows that with probability not less than $1 - 3m_\Sigma m_\Lambda e^{-\tau}$ we have*

$$\mathcal{R}_{L,P}(\hat{f}_{T_0,\sigma,\lambda}) - \mathcal{R}^*_{L,P} \le K_{d,q,\alpha,\varepsilon}\left(\left(\frac{\sigma^d}{\lambda^\varepsilon n}\right)^{\frac{q+1}{q+2-\varepsilon}} + \left(\frac{\tau}{n}\right)^{\frac{q+1}{q+2}} + \sigma^d \lambda + \sigma^{-\alpha d}\right)$$

simultaneously *for all $\sigma \in \Sigma$ and $\lambda \in \Lambda$, where $\varepsilon \in (0, 1]$ is arbitrarily but fixed and $K_{d,q,\alpha,\varepsilon}$ is a suitable constant. Now using a simple model selection approach (see e.g. Theorem 4.2) for the second half $T_1$ of our training set we find that with probability not less than $1 - e^{-\tau}$ we have*

$$\mathcal{R}_{L,P}(\hat{f}_{T_0,\sigma^*_{T_1},\lambda^*_{T_1}}) - \mathcal{R}^*_{L,P} \ \le \ C\left(\frac{\tau + \log(m_\Sigma m_\Lambda)}{n}\right)^{\frac{q+1}{q+2}}$$

$$+ C \min_{\sigma \in \Sigma, \lambda \in \Lambda}\left(\left(\frac{\sigma^d}{\lambda^\varepsilon n}\right)^{\frac{q+1}{q+2-\epsilon}} + \sigma^d \lambda + \sigma^{-\alpha d}\right),$$

*where $C$ is a constant only depending on $d$, $q$, $\alpha$, and $\varepsilon$, and $(\sigma^*_{T_1}, \lambda^*_{T_1}) \in \Sigma \times \Lambda$ is a pair that minimizes the empirical risk $\mathcal{R}_{L,T_1}(.)$ over $\Sigma \times \Lambda$.*
*Now assume that $\Sigma_n$ and $\Lambda_n$ are $1/n$- and $1/n^2$-nets of $[1, n^{1/d})$ and $(0, 1]$, respectively. Obviously, we can choose $\Sigma_n$ and $\Lambda_n$ such that $m_{\Sigma_n} \le n^2$ and $m_{\Lambda_n} \le n^2$, respectively. With such parameter sets it is then easy to check that we obtain exactly the rates we have found in Example 2.3, but without knowing the noise exponents $\alpha$ and $q$ a-priori.*

## 3  An oracle inequality for clipped penalized ERM

Theorem 2.1 is a consequence of a far more general oracle inequality on clipped penalized empirical risk minimizers. Since this result is of its own interest we now present it together with its proof in detail. To this end recall that a subroot is a nondecreasing function $\varphi : [0, \infty) \to [0, \infty)$ such that $\varphi(r)/\sqrt{r}$ is nonincreasing in $r$. Moreover, for a Rademacher sequence $\sigma := (\sigma_1, \ldots, \sigma_n)$ with respect to the measure $\nu$ and a function $h : Z \to \mathbb{R}$ we define $R_\sigma h : Z^n \to \mathbb{R}$ by $R_\sigma h := n^{-1}(\sigma_1 h(z_1) + \cdots + \sigma_n h(z_n))$. Now the general oracle inequality is:

**Theorem 3.1** *Let $\mathcal{P} \neq \emptyset$ be a set of (hyper)-parameters, $\mathcal{F}$ be a set of measurable functions $f : X \to \mathbb{R}$ with $0 \in \mathcal{F}$, and $\Omega : \mathcal{P} \times \mathcal{F} \to [0, \infty]$ be a function. Let $P$ be a distribution on $X \times Y$ and $L$ be a loss function which satisfies (3) and (4). For a fixed pair $(p_0, f_0) \in \mathcal{P} \times \mathcal{F}$ we define*

$$a_\Omega(p_0, f_0) := \Omega(p_0, f_0) + \mathcal{R}_{L,P}(f_0) - \mathcal{R}^*_{L,P}.$$

*Moreover, let us assume that the quantity $B(f_0)$ defined in (5) is finite. In addition, we assume that we have a variance bound of the form (6) for constants $v \geq 1$, $\vartheta \in [0, 1]$ and all measurable $f : X \to \mathbb{R}$. Furthermore, suppose that there exists a subroot $\varphi_n$ with*

$$\mathbb{E}_{T \sim P^n} \mathbb{E}_{\sigma \sim \nu} \sup_{\substack{(p,f) \in \mathcal{P} \times \mathcal{F} \\ \Omega(p,f) + \mathbb{E}_P(L \circ \hat{f} - L \circ f^*_{L,P}) \leq r}} \left| R_\sigma(L \circ \hat{f} - L \circ f^*_{L,P}) \right| \leq \varphi_n(r), \qquad r > 0. \quad (9)$$

*Finally, let $(p_{T,\Omega}, f_{T,\Omega})$ be an $\epsilon$-approximate minimizer of $(p, f) \mapsto \Omega(p, f) + \mathcal{R}_{L,T}(f)$. Then for all $\tau \geq 1$ and all $r$ satisfying*

$$r \geq \max\left\{ 120\varphi_n(r), \left(\frac{32v\tau}{n}\right)^{\frac{1}{2-\vartheta}}, \frac{28\tau}{n} \right\} \quad (10)$$

*we have with probability not less than $1 - 3e^{-\tau}$ that*

$$\Omega(p_{T,\Omega}, f_{T,\Omega}) + \mathcal{R}_{L,P}(\hat{f}_{T,\Omega}) - \mathcal{R}^*_{L,P} \leq 5r + \frac{14B(f_0)\tau}{3n} + 8a_\Omega(p_0, f_0) + 4\epsilon.$$

***Proof:*** We write $B$ for $B(f_0)$. For $T \in (X \times Y)^n$ we now observe $\Omega(p_{T,\Omega}, f_{T,\Omega}) + \mathcal{R}_{L,T}(\hat{f}_{T,\Omega}) - \Omega(p_0, f_0) - \mathcal{R}_{L,T}(f_0) \leq \epsilon$ by the definition of $(p_{T,\Omega}, f_{T,\Omega})$, and hence we find

$$\Omega(p_{T,\Omega}, f_{T,\Omega}) + \mathcal{R}_{L,P}(\hat{f}_{T,\Omega}) - \mathcal{R}^*_{L,P}$$
$$\leq \mathcal{R}_{L,P}(\hat{f}_{T,\Omega}) - \mathcal{R}_{L,T}(\hat{f}_{T,\Omega}) + \mathcal{R}_{L,T}(f_0) - \mathcal{R}_{L,P}(f_0) + a_\Omega(p_0, f_0) + \epsilon$$
$$= \mathcal{R}_{L,P}(\hat{f}_{T,\Omega}) - \mathcal{R}_{L,P}(f^*_{L,P}) - \mathcal{R}_{L,T}(\hat{f}_{T,\Omega}) + \mathcal{R}_{L,T}(f^*_{L,P}) \quad (11)$$
$$+ \mathcal{R}_{L,T}(f_0) - \mathcal{R}_{L,T}(\hat{f}_0) - \mathcal{R}_{L,P}(f_0) + \mathcal{R}_{L,P}(\hat{f}_0) \quad (12)$$
$$+ \mathcal{R}_{L,T}(\hat{f}_0) - \mathcal{R}_{L,T}(f^*_{L,P}) - \mathcal{R}_{L,P}(\hat{f}_0) + \mathcal{R}_{L,P}(f^*_{L,P}) \quad (13)$$
$$+ a_\Omega(p_0, f_0) + \epsilon.$$

Let us first estimate the term in line (12). To this end we write $h_1 := L \circ f_0 - L \circ \hat{f}_0$. Then our assumption on $L$ guarantees $h_1 \geq 0$, and since we also have $\|h_1\|_\infty \leq B$, we find $\|h_1 - \mathbb{E}_P h_1\|_\infty \leq B$. In addition, we obviously have $\mathbb{E}_P(h_1 - \mathbb{E}_P h_1)^2 \leq \mathbb{E}_P h_1^2 \leq B\mathbb{E}_P h_1$. Consequently, Bernstein's inequality [6, Thm. 8.2] shows that with probability not less than $1 - e^{-\tau}$ we have

$$\mathbb{E}_T h_1 - \mathbb{E}_P h_1 < \sqrt{\frac{2\tau B \, \mathbb{E}_P h_1}{n}} + \frac{2B\tau}{3n}.$$

Now using $\sqrt{ab} \leq \frac{a}{2} + \frac{b}{2}$ we find $\sqrt{2\tau B \mathbb{E}_P h_1} \cdot n^{-\frac{1}{2}} \leq \mathbb{E}_P h_1 + \frac{B\tau}{2n}$, and consequently we have

$$P^n\left( T \in Z^n : \mathcal{R}_{L,T}(f_0) - \mathcal{R}_{L,T}(\hat{f}_0) - \mathcal{R}_{L,P}(f_0) + \mathcal{R}_{L,P}(\hat{f}_0) < \mathbb{E}_P h_1 + \frac{7B\tau}{6n} \right) \geq 1 - e^{-\tau}. \quad (14)$$

Let us now estimate the term in line (13). To this end we write $h_2 := L \circ \hat{f}_0 - L \circ f^*_{L,P}$. Then we have $\|h_2\|_\infty \leq 3$ and $\|h_2 - \mathbb{E}_P h_2\|_\infty \leq 6$. In addition, our variance bound gives $\mathbb{E}_P(h_2 - \mathbb{E}_P h_2)^2 \leq \mathbb{E}_P h_2^2 \leq v(\mathbb{E}_P h_2)^\vartheta$, and consequently, Bernstein's inequality shows that with probability not less than $1 - e^{-\tau}$ we have

$$\mathbb{E}_T h_2 - \mathbb{E}_P h_2 < \sqrt{\frac{2\tau v(\mathbb{E}_P h_2)^\vartheta}{n}} + \frac{4\tau}{n}.$$

Now, for $q^{-1} + (q')^{-1} = 1$ the elementary inequality $ab \leq a^q q^{-1} + b^{q'}(q')^{-1}$ holds, and hence for $q := \frac{2}{2-\vartheta}$, $q' := \frac{2}{\vartheta}$, $a := \sqrt{2^{1-\vartheta}\vartheta^\vartheta \tau v} \cdot n^{-\frac{1}{2}}$, and $b := \left(\frac{2\mathbb{E}_P h_2}{\vartheta}\right)^{\vartheta/2}$ we obtain

$$\sqrt{\frac{2\tau v(\mathbb{E}_P h_2)^\vartheta}{n}} \leq \left(1 - \frac{\vartheta}{2}\right)\left(\frac{2^{1-\vartheta}\vartheta^\vartheta v\tau}{n}\right)^{\frac{1}{2-\vartheta}} + \mathbb{E}_P h_2.$$

Since elementary calculations show that $\left(2^{-\vartheta}\vartheta^{\vartheta}\right)^{\frac{1}{2-\vartheta}} \leq 1$ we obtain

$$\sqrt{\frac{2\tau v(\mathbb{E}_P h_2)^{\vartheta}}{n}} \leq \left(1 - \frac{\vartheta}{2}\right)\left(\frac{2v\tau}{n}\right)^{\frac{1}{2-\vartheta}} + \mathbb{E}_P h_2.$$

Therefore we have with probability not less than $1 - e^{-\tau}$ that

$$\mathcal{R}_{L,T}(\hat{f}_0) - \mathcal{R}_{L,T}(f_{L,P}^*) - \mathcal{R}_{L,P}(\hat{f}_0) + \mathcal{R}_{L,P}(f_{L,P}^*) < \mathbb{E}_P h_2 + \left(1 - \frac{\vartheta}{2}\right)\left(\frac{2v\tau}{n}\right)^{\frac{1}{2-\vartheta}} + \frac{4\tau}{n}. \quad (15)$$

Let us finally estimate the term in line (11). To this end we write $h_f := L \circ \hat{f} - L \circ f_{L,P}^*$, $f \in \mathcal{F}$. Moreover, for $r > 0$ we define

$$\mathcal{G}_r := \left\{ \frac{\mathbb{E}_P h_f - h_f}{\Omega(p,f) + \mathbb{E}_P(h_f) + r} : (p,f) \in \mathcal{P} \times \mathcal{F}\right\}.$$

Then for $g_{p,f} := \frac{\mathbb{E}_P h_f - h_f}{\Omega(p,f) + \mathbb{E}_P(h_f) + r} \in \mathcal{G}_r$ we have $\mathbb{E}_P g_{p,f} = 0$ and

$$\|g_{p,f}\|_{\infty} = \sup_{z \in Z}\left|\frac{\mathbb{E}_P h_f - h_f(z)}{\Omega(p,f) + \mathbb{E}_P(h_f) + r}\right| = \frac{\|\mathbb{E}_P h_f - h_f\|_{\infty}}{\Omega(p,f) + \mathbb{E}_P(h_f) + r} \leq \frac{6}{r}.$$

In addition, the inequality $a^{\vartheta} b^{2-\vartheta} \leq (a + b)^2$ and the variance bound assumption (6) implies that

$$\mathbb{E}_P g_{p,f}^2 \leq \frac{\mathbb{E}_P h_f^2}{(\mathbb{E}_P(h_f) + r)^2} \leq \frac{\mathbb{E}_P h_f^2}{r^{2-\vartheta}(\mathbb{E}_P h_f)^{\vartheta}} \leq \frac{v}{r^{2-\vartheta}}.$$

Now define

$$\Phi(r) := \mathbb{E}_{T \sim P^n} \sup_{(p,f) \in \mathcal{P} \times \mathcal{F}} \frac{\mathbb{E}_P h_f - \mathbb{E}_T h_f}{\Omega(p,f) + \mathbb{E}_P(h_f) + r}.$$

Standard symmetrization then yields

$$\mathbb{E}_{T \sim P^n} \sup_{\substack{(p,f) \in \mathcal{P} \times \mathcal{F} \\ \Omega(p,f) + \mathbb{E}_P(h_f) \leq r}} |\mathbb{E}_P h_f - \mathbb{E}_T h_f| \leq 2\mathbb{E}_{T \sim P^n}\mathbb{E}_{\sigma \sim \nu} \sup_{\substack{(p,f) \in \mathcal{P} \times \mathcal{F} \\ \Omega(p,f) + \mathbb{E}_P(h_f) \leq r}} |R_{\sigma} h_f|,$$

and hence Lemma 3.2 proved below together with (9) shows $\Phi(r) \leq 10\varphi_n(r)r^{-1}$, $r > 0$. Therefore applying Talagrand's inequality in the version of [3] to the class $\mathcal{G}_r$ we obtain

$$P^n\left(T \in Z^n : \sup_{g \in \mathcal{G}_r} \mathbb{E}_T g \leq \frac{30\varphi_n(r)}{r} + \sqrt{\frac{2\tau v}{nr^{2-\vartheta}}} + \frac{7\tau}{nr}\right) \geq 1 - e^{-\tau}.$$

Let us define $\varepsilon_r := \frac{30\varphi_n(r)}{r} + \left(\frac{2\tau v}{nr^{2-\vartheta}}\right)^{1/2} + \frac{7\tau}{nr}$. Then the above inequality gives with probability not less than $1 - e^{-\tau}$ that for all $(p,f) \in \mathcal{P} \times \mathcal{F}$ we have

$$\mathbb{E}_P h_f - \mathbb{E}_T h_f \leq \varepsilon_r \cdot \left(\Omega(p,f) + \mathbb{E}_P h_f\right) + 30\varphi_n(r) + \sqrt{\frac{2\tau v r^{\vartheta}}{n}} + \frac{7\tau}{n},$$

and consequently we have with probability not less than $1 - e^{-\tau}$ that

$$\mathcal{R}_{L,P}(\hat{f}_{T,\Omega}) - \mathcal{R}_{L,P}(f_{L,P}^*) - \mathcal{R}_{L,T}(\hat{f}_{T,\Omega}) + \mathcal{R}_{L,T}(f_{L,P}^*)$$

$$\leq \varepsilon_r \cdot \left(\Omega(p_{T,\Omega}, f_{T,\Omega}) + \mathcal{R}_{L,P}(\hat{f}_{T,\Omega}) - \mathcal{R}_{L,P}(f_{L,P}^*)\right) + 30\varphi_n(r) + \sqrt{\frac{2\tau v r^{\vartheta}}{n}} + \frac{7\tau}{n}. \quad (16)$$

Now observe that for the functions $h_1$ and $h_2$ which we defined when estimating (12) and (13) we have

$$\mathbb{E}_P g + \mathbb{E}_P h = \mathcal{R}_{L,P}(f_0) - \mathcal{R}_{L,P}^*, \quad (17)$$

and hence we can combine our estimates (16), (14), and (15) of the terms (11), (12), and (13) to obtain that with probability not less than $1 - 3e^{-\tau}$ we have

$$(1 - \varepsilon_r)\left(\Omega(p_{T,\Omega}, f_{T,\Omega}) + \mathcal{R}_{L,P}(\hat{f}_{T,\Omega}) - \mathcal{R}_{L,P}^*\right)$$

$$\leq 30\varphi_n(r) + \sqrt{\frac{2\tau v r^{\vartheta}}{n}} + \left(1 - \frac{\vartheta}{2}\right)\left(\frac{2v\tau}{n}\right)^{\frac{1}{2-\vartheta}} + \frac{(66 + 7B)\tau}{6n} + a_{\Omega}(p_0, f_0) + \mathcal{R}_{L,P}(f_0) - \mathcal{R}_{L,P}^* + \epsilon.$$

In particular, for $r$ satisfying the assumption (10) we have $\frac{30\varphi_n(r)}{r} \leq \frac{1}{4}$, $\left(\frac{2\tau v}{nr^{2-\vartheta}}\right)^{1/2} \leq \frac{1}{4}$, and $\frac{7\tau}{nr} \leq \frac{1}{4}$. This shows $1 - \varepsilon_r \geq \frac{1}{4}$, and hence we obtain with probability not less than $1 - 3e^{-\tau}$ that

$$\Omega(p_{T,\Omega}, f_{T,\Omega}) + \mathcal{R}_{L,P}(\hat{f}_{T,\Omega}) - \mathcal{R}_{L,P}^* \leq 120\varphi_n(r) + \sqrt{\frac{32\tau vr^\vartheta}{n}} + 2(2-\vartheta)\left(\frac{2v\tau}{n}\right)^{\frac{1}{2-\vartheta}} + \frac{44\tau}{n}$$
$$+ \frac{14B\tau}{3n} + 4a_\Omega(p_0, f_0) + 4\left(\mathcal{R}_{L,P}(f_0) - \mathcal{R}_{L,P}^*\right) + 4\epsilon.$$

However we also have $120\varphi_n(r) \leq r$, $\left(\frac{32\tau vr^\vartheta}{n}\right)^{1/2} \leq r$, $\frac{44\tau}{n} \leq \frac{5r}{3}$, and $2(2-\vartheta)\left(\frac{2v\tau}{n}\right)^{\frac{1}{2-\vartheta}} \leq 2(2-\vartheta)\frac{r}{4} \leq r$, and hence we find the assertion. ■

For the proof of Theorem 3.1 it remains to show the following lemma:

**Lemma 3.2** *Let $\mathcal{P}$ and $\mathcal{F}$ be as in Theorem 3.1. Furthermore, let $W : \mathcal{F} \to \mathbb{R}$ and $a : \mathcal{P} \times \mathcal{F} \to [0, \infty)$. Define*

$$\Phi(r) := \mathbb{E}_{T \sim P^n} \sup_{f \in \mathcal{P} \times \mathcal{F}} \frac{|\mathbb{E}_T W(f) - \mathbb{E}_P W(f)|}{a(p, f) + r}$$

*and suppose that there exists a subroot $\Psi$ such that*

$$\mathbb{E}_{T \sim P^n} \sup_{\substack{(p,f) \in \mathcal{P} \times \mathcal{F} \\ a(p,f) \leq r}} \left|\mathbb{E}_T W(f) - \mathbb{E}_P W(f)\right| \leq \Psi(r), \qquad r > 0.$$

*Then we have $\Phi(r) \leq \frac{5}{r}\Psi(r)$ for all $r > 0$.*

**Proof:** For $x > 1$, $r > 0$, and $T \in (X \times Y)^n$ we obtain by a standard peeling approach that

$$\sup_{(p,f) \in \mathcal{P} \times \mathcal{F}} \frac{|\mathbb{E}_P W(f) - \mathbb{E}_T W(f)|}{a(p, f) + r}$$

$$\leq \sup_{\substack{(p,f) \in \mathcal{P} \times \mathcal{F} \\ a(p,f) \leq r}} \frac{|\mathbb{E}_P W(f) - \mathbb{E}_T W(f)|}{a(p, f) + r} + \sum_{i=0}^{\infty} \sup_{\substack{(p,f) \in \mathcal{P} \times \mathcal{F} \\ a(p,f) \geq rx^i \\ a(p,f) \leq rx^{i+1}}} \frac{|\mathbb{E}_P W(f) - \mathbb{E}_T W(f)|}{a(p, f) + r}$$

$$\leq \sup_{\substack{(p,f) \in \mathcal{P} \times \mathcal{F} \\ a(p,f) \leq r}} \frac{|\mathbb{E}_P W(f) - \mathbb{E}_T W(f)|}{r} + \sum_{i=0}^{\infty} \sup_{\substack{(p,f) \in \mathcal{P} \times \mathcal{F} \\ a(p,f) \geq rx^i \\ a(p,f) \leq rx^{i+1}}} \frac{|\mathbb{E}_P W(f) - \mathbb{E}_T W(f)|}{rx^i + r}$$

$$\leq \frac{1}{r} \sup_{\substack{(p,f) \in \mathcal{P} \times \mathcal{F} \\ a(p,f) \leq r}} |\mathbb{E}_P W(f) - \mathbb{E}_T W(f)| + \frac{1}{r}\sum_{i=0}^{\infty} \frac{1}{x^i + 1} \sup_{\substack{(p,f) \in \mathcal{P} \times \mathcal{F} \\ a(p,f) \leq rx^{i+1}}} |\mathbb{E}_P W(f) - \mathbb{E}_T W(f)|$$

$$= \frac{1}{r}\left(\Psi(r) + \sum_{i=0}^{\infty} \frac{\Psi(rx^{i+1})}{x^i + 1}\right).$$

However since $\Psi$ is a subroot we obtain that $\Psi(rx^{i+1}) \leq x^{\frac{i+1}{2}}\Psi(r)$ so that we obtain the assertion by setting $x := 4$. ■

## 4   Proof of Theorem 2.1

Before we begin the proof of Theorem 2.1 let us state the following proposition which follows directly from [8] (see also [9, Prop. 5.7]) together with simple considerations on covering numbers:

**Proposition 4.1** *Let $\mathcal{F} := H$ be a RKHS, $\mathcal{P} := \{p_0\}$ be a singleton, and $\Omega(p_0, f) := \lambda\|f\|^2$. If (7) is satisfied then there exists a constant $c_p$ depending only on $p$ such that (9) is satisfied for*

$$\varphi_n(r) := c_p \max\left\{v^{\frac{1}{2}(1-p)}r^{\frac{\vartheta}{2}(1-p)}\left(\frac{r}{\lambda}\right)^{\frac{p}{2}}\left(\frac{a}{n}\right)^{\frac{1}{2}}, \left(\frac{r}{\lambda}\right)^{\frac{p}{1+p}}\left(\frac{a}{n}\right)^{\frac{1}{1+p}}\right\}.$$

***Proof of Theorem 2.1:*** From the covering bound assumption we observe that Proposition 4.1 implies we have the bound (9) with $\varphi_n(r)$ defined by the righthand side of Proposition 4.1 and therefore Theorem 3.1 implies that Condition (10) becomes

$$r \geq \max\left\{ 120c_p v^{\frac{1}{2}(1-p)} r^{\frac{\vartheta}{2}(1-p)} \left(\frac{r}{\lambda}\right)^{\frac{p}{2}} \left(\frac{a}{n}\right)^{\frac{1}{2}}, 120c_p \left(\frac{r}{\lambda}\right)^{\frac{p}{1+p}} \left(\frac{a}{n}\right)^{\frac{1}{1+p}}, \left(\frac{32v\tau}{n}\right)^{\frac{1}{2-\vartheta}}, \frac{28\tau}{n} \right\} \quad (18)$$

and solving with respect to $r$ yields the conclusion. ∎

Finally, for the parameter selection approach in Example 2.4 we need the following oracle inequality for model selection:

**Theorem 4.2** *Let $P$ be a distribution on $X \times Y$ and let $L$ be a loss function which satisfies (3), (4), and the variance bound (6). Furthermore, let $\mathcal{F} := \{f_1, \ldots, f_m\}$ be a finite set of functions mapping $X$ into $[-1, 1]$. For $T \in (X \times Y)^n$ we define*

$$f_T := \arg\min_{f \in \mathcal{F}} \mathcal{R}_{L,T}(f) \,.$$

*Then there exists a universal constant $K$ such that for all $\tau \geq 1$ we have with probability not less than $1 - 3e^{-\tau}$ that*

$$\mathcal{R}_{L,P}(f_T) - \mathcal{R}^*_{L,P} \leq 5\left(\frac{K\log m}{n}\right)^{\frac{1}{2-\vartheta}} + 5\left(\frac{32v\tau}{n}\right)^{\frac{1}{2-\vartheta}} + \frac{5K\log m + 154\tau}{n}$$

$$+ 8\min_{f \in \mathcal{F}}(\mathcal{R}_{L,P}(f) - \mathcal{R}^*_{L,P}) \,.$$

***Proof:*** Since all functions $f_i$ already map into $[-1, 1]$ we do not have to consider the clipping operator. For $r > 0$ we now define $\mathcal{F}_r := \{f \in \mathcal{F} : \mathcal{R}_{L,P}(f) - \mathcal{R}^*_{L,P} \leq r\}$. Then the cardinality of $\mathcal{F}_r$ is smaller than or equal to $m$ and hence we have $\mathcal{N}(L \circ \mathcal{F}_r - L \circ f^*_{L,P}, \varepsilon, L_2(T)) \leq m$ for all $\varepsilon > 0$. Using the technique of [8] (cf. also [9, Prop. 5.7]) we hence obtain that (9) is satisfied for

$$\varphi_n(r) := \frac{c}{\sqrt{n}} \max\left\{ \sqrt{v\log m}\, r^{\vartheta/2}, \frac{\log m}{\sqrt{n}} \right\},$$

where $c$ is a universal constant. Applying Theorem 3.1 then yields the assertion. ∎

# References

[1] P.L. Bartlett. The sample complexity of pattern classification with neural networks: the size of the weights is more important than the size of the network. *IEEE Trans. Inform. Theory*, 44:525–536, 1998.

[2] G. Blanchard, O. Bousquet, and P. Massart. Statistical performance of support vector machines. Technical Report, 2004.

[3] O. Bousquet. A Bennet concentration inequality and its application to suprema of empirical processes. *C. R. Math. Acad. Sci. Paris*, 334:495–500, 2002.

[4] D.R. Chen, Q. Wu, Y.M. Ying, and D.X. Zhou. Support vector machine soft margin classifiers: Error analysis. *Journal of Machine Learning Research*, 5:1143–1175, 2004.

[5] N. Cristianini and J. Shawe-Taylor. *An Introduction to Support Vector Machines*. Cambridge University Press, 2000.

[6] L. Devroye, L. Györfi, and G. Lugosi. *A Probabilistic Theory of Pattern Recognition*. Springer, New York, 1996.

[7] F. Girosi, M. Jones, and T. Poggio. Regularization theory and neural networks architectures. *Neural Computation*, 7:219–269, 1995.

[8] S. Mendelson. Improving the sample complexity using global data. *IEEE Trans. Inform. Theory*, 48:1977–1991, 2002.

[9] I. Steinwart and C. Scovel. Fast rates for support vector machines using Gaussian kernels. *Annals of Statistics*, to appear.

[10] I. Steinwart and C. Scovel. Fast rates for support vector machines. In *Proceedings of the 18th Annual Conference on Learning Theory, COLT 2005*, pages 279–294. Springer, 2005.

[11] Q. Wu, Y. Ying, and D.-X. Zhou. Multi-kernel regularized classifiers. *J. Complexity*, to appear.
